# Learning Auto-regressive Models from Sequence and Non-sequence Data

**Tzu-Kuo Huang**
Machine Learning Department
Carnegie Mellon University
tzukuoh@cs.cmu.edu

**Jeff Schneider**
Robotics Institute
Carnegie Mellon University
schneide@cs.cmu.edu

## Abstract

Vector Auto-regressive models (VAR) are useful tools for analyzing time series data. In quite a few modern time series modelling tasks, the collection of reliable time series turns out to be a major challenge, either due to the slow progression of the dynamic process of interest, or inaccessibility of repetitive measurements of the same dynamic process over time. In those situations, however, we observe that it is often easier to collect a large amount of non-sequence samples, or snapshots of the dynamic process of interest. In this work, we assume a small amount of time series data are available, and propose methods to incorporate non-sequence data into penalized least-square estimation of VAR models. We consider non-sequence data as samples drawn from the stationary distribution of the underlying VAR model, and devise a novel penalization scheme based on the Lyapunov equation concerning the covariance of the stationary distribution. Experiments on synthetic and video data demonstrate the effectiveness of the proposed methods.

## 1 Introduction

Vector Auto-regressive models (VAR) are an important class of models for analyzing multivariate time series data. They have proven to be very useful in capturing and forecasting the dynamic properties of time series in a number of domains, such as finance and economics [18, 13]. Recently, researchers in computational biology applied VAR models in the analysis of genomic time series [12], and found interesting results that were unknown previously.

In quite a few scientific modeling tasks, a major difficulty turns out to be the collection of reliable time series data. In some situations, the dynamic process of interest may evolve slowly over time, such as the progression of Alzheimer's or Parkinson's diseases, and researchers may need to spend months or even years tracking the dynamic process to obtain enough time series data for analysis. In other situations, the dynamic process of interest may not be able to undergo repetitive measurements, so researchers have to measure multiple instances of the same process while maintaining synchronization among these instances. One such example is gene expression time series. In their study, [19] measured expression profiles of yeast genes along consecutive metabolic cycles. Due to the destructive nature of the measurement technique, they collected expression data from multiple yeast cells. In order to obtain reliable time series data, they spent a lot of effort developing a stable environment to synchronize the cells during the metabolic cycles. Yet, they point out in their discussion that such a synchronization scheme may not work for other species, e.g., certain bacteria and fungi, as effectively as for yeast.

While obtaining reliable time series can be difficult, we observe that it is often easier to collect non-sequence samples, or snapshots of the dynamic process of interest[1]. For example, a scientist studying

Alzheimer's or Parkinson's can collect samples from his or her current pool of patients, each of whom may be in a different stage of the disease. Or in gene expression analysis, current technology already enables large-scale collection of static gene expression data. Previously [6] investigated ways to extract dynamics from such static gene expression data, and more recently [8, 9] proposed methods for learning first-order dynamic models from general non-sequence data. However, most of these efforts suffer from a fundamental limitation: due to lack of temporal information, multiple dynamic models may fit the data equally well and hence certain characteristics of dynamics, such as the step size of a discrete-time model and the overall temporal direction, become non-identifiable.

In this work, we aim to combine these two types of data to improve learning of dynamic models. We assume that a small amount of sequence samples and a large amount of non-sequence samples are available. Our aim is to rely on the few sequence samples to obtain a rough estimate of the model, while refining this rough estimate using the non-sequence samples. We consider the following first-order $p$-dimensional vector auto-regressive model:

$$\mathbf{x}^{t+1} = \mathbf{x}^t A + \boldsymbol{\epsilon}^{t+1}, \tag{1}$$

where $\mathbf{x}^t \in \mathbb{R}^{1 \times p}$ is the state vector at time $t$, $A \in \mathbb{R}^{p \times p}$ is the transition matrix, and $\boldsymbol{\epsilon}^t$ is a white-noise process with a time-invariant variance $\sigma^2 I$. Given a sequence sample, a common estimation method for $A$ is the least-square estimator, whose properties have been studied extensively (see e.g., [7]). We assume that the process (1) is stable, i.e., the eigenvalues of $A$ have modulus less than one. As a result, the process (1) has a stationary distribution, whose covariance $Q$ is determined by the following discrete-time Lyapunov equation:

$$A^\top Q A + \sigma^2 I = Q. \tag{2}$$

Linear quadratic Lyapunov theory (see e.g., [1]) gives that $Q$ is *uniquely* determined if and only if $\lambda_i(A)\lambda_j(A) \neq 1$ for $1 \leq i, j \leq p$, where $\lambda_i(A)$ is the $i$-th eigenvalue of $A$. If the noise process $\boldsymbol{\epsilon}^t$ follows a normal distribution, the stationary distribution also follows a normal distribution, with covariance $Q$ determined as above. Since our goal is to estimate $A$, a more relevant perspective is viewing (2) as a system of constraints on $A$. What motivates this work is that the estimation of $Q$ requires only samples drawn from the stationary distribution rather than sequence data. However, even if we have the true $Q$ and $\sigma^2$, we still cannot uniquely determine $A$ because (2) is an under-determined system[2] of $A$. We thus rely on the few sequence samples to resolve the ambiguity.

We describe the proposed methods in Section 2, and demonstrate their performance through experiments on synthetic and video data in Section 3. Our finding in short is that when the amount of sequence data is small and our VAR model assumption is valid, the proposed methods of incorporating non-sequence data into estimation significantly improve over standard methods, which use only the sequence data. We conclude this work and discuss future directions in Section 4.

## 2 Proposed Methods

Let $\{\mathbf{x}^i\}_{i=1}^T$ be a sequence of observations generated by the process (1). The standard least-square estimator for the transition matrix $A$ is the solution to the following minimization problem:

$$\min_A \quad \|Y - XA\|_F^2, \tag{3}$$

where $Y^\top := [(\mathbf{x}^2)^\top \ (\mathbf{x}^3)^\top \cdots (\mathbf{x}^T)^\top]$, $X^\top := [(\mathbf{x}^1)^\top \ (\mathbf{x}^2)^\top \cdots (\mathbf{x}^{T-1})^\top]$, and $\| \cdot \|_F$ denotes the matrix Frobenius norm. When $p > T$, which is often the case in modern time series modeling tasks, the least square problem (3) has multiple solutions all achieving zero squared error, and the resulting estimator overfits the data. A common remedy is adding a penalty term on $A$ to (3) and minimizing the resulting regularized sum of squared errors. Usual penalty terms include the ridge penalty $\|A\|_F^2$ and the sparse penalty $\|A\|_1 := \sum_{i,j} |A_{ij}|$.

Now suppose we also have a set of non-sequence observations $\{\mathbf{z}_i\}_{i=1}^n$ drawn independently from the stationary distribution of (1). Note that we use superscripts for time indices and subscripts for data indices. As described in Section 1, the size $n$ of the non-sequence sample can usually be much larger than the size $T$ of the sequence data. To incorporate the non-sequence observations into the

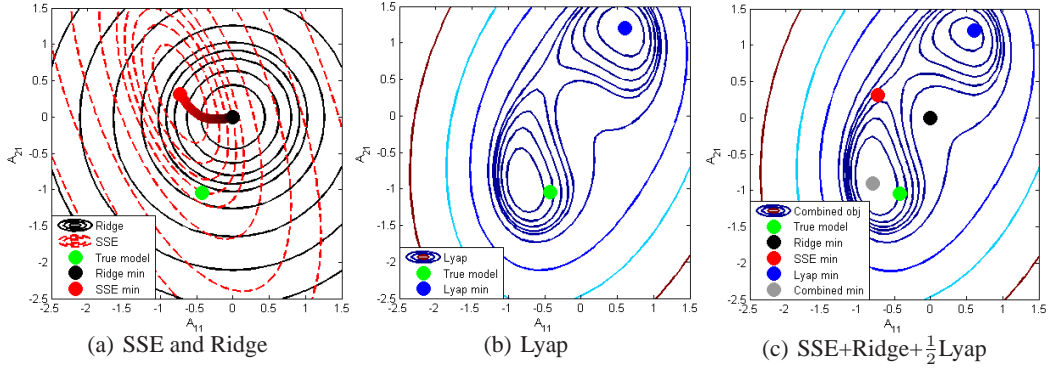

(a) SSE and Ridge          (b) Lyap          (c) SSE+Ridge+$\frac{1}{2}$Lyap

Figure 1: Level sets of different functions in a bivariate AR example

estimation procedure, we first obtain a covariance estimate $\widehat{Q}$ of the stationary distribution from the non-sequence sample, and then turn the Lyapunov equation (2) into a regularization term on $A$. More precisely, in addition to the usual ridge or sparse penalty terms, we also consider the following regularization:

$$\|A^\top \widehat{Q} A + \sigma^2 I - \widehat{Q}\|_F^2, \tag{4}$$

which we refer to as the *Lyapunov penalty*. To compare (4) with the ridge penalty and the sparse penalty, we consider (3) as a multiple-response regression problem and view the $i$-th column of $A$ as the regression coefficient vector for the $i$-th output dimension. From this viewpoint, we immediately see that both the ridge and the sparse penalizations treat the $p$ regression problems as unrelated. On the contrary, the Lyapunov penalty incorporates relations between pairs of columns of $A$ by using a covariance estimate $\widehat{Q}$. In other words, although the non-sequence sample does not provide direct information about the individual regression problems, it does reveal how the regression problems are related to one another. To illustrate how the Lyapunov penalty may help to improve learning, we give an example in Figure 1. The true transition matrix is

$$A = \begin{bmatrix} -0.4280 & 0.5723 \\ -1.0428 & -0.7144 \end{bmatrix} \tag{5}$$

and $\epsilon^t \sim \mathcal{N}(\mathbf{0}, I)$. We generate a sequence of 4 points, draw a non-sequence sample of 20 points independently from the stationary distribution and obtain the sample covariance $\widehat{Q}$. We fix the second column of $A$ but vary the first, and plot in Figure 1(a) the resulting level sets of the sum of squared errors on the sequence (SSE) and the ridge penalty (Ridge), and in Figure 1(b) the level sets of the Lyapunov penalty (Lyap). We also give coordinates of the true $[A_{11}\ A_{21}]^\top$, the minima of SSE, Ridge, and Lyap, respectively. To see the behavior of the ridge regression, we trace out a path of the ridge regression solution by varying the penalization parameter, as indicated by the red-to-black curve in Figure 1(a). This path is pretty far from the true model, due to insufficient sequence data. For the Lyapunov penalty, we observe that it has two local minima, one of which is very close to the true model, while the other, also the global minimum, is very far. Thus, neither ridge regression nor the Lyapunov penalty can be used on its own to estimate the true model well. But as shown in Figure 1(c), the combined objective, SSE+Ridge+$\frac{1}{2}$Lyap, has its global minimum very close to the true model. This demonstrates how the ridge regression and the Lyapunov penalty may complement each other: the former by itself gives an inaccurate estimation of the true model, but is just enough to identify a good model from the many candidate local minima provided by the latter.

In the following we describe our proposed methods for incorporating the Lyapunov penalty (4) into ridge and sparse least-square estimation. We also discuss robust estimation for the covariance $Q$.

## 2.1   Ridge and Lyapunov penalty

Here we estimate $A$ by solving the following problem:

$$\min_A \quad \frac{1}{2}\|Y - XA\|_F^2 + \frac{\lambda_1}{2}\|A\|_F^2 + \frac{\lambda_2}{4}\|A^\top \widehat{Q} A + \sigma^2 I - \widehat{Q}\|_F^2, \tag{6}$$

where $\widehat{Q}$ is a covariance estimate obtained from the non-sequence sample. We treat $\lambda_1$, $\lambda_2$ and $\sigma^2$ as hyperparameters and determine their values on a validation set. Given these hyperparameters, we solve (6) by gradient descent with back-tracking line search for the step size. The gradient of the objective function is given by

$$-X^\top Y + X^\top X A + \lambda_1 A + \lambda_2 \widehat{Q} A (A^\top \widehat{Q} A + \sigma^2 I - \widehat{Q}). \tag{7}$$

As mentioned before, (6) is a non-convex problem and thus requires good initialization. We use the following two initial estimates of $A$:

$$\widehat{A}^{lsq} := (X^\top X)^\dagger X^\top Y \quad \text{and} \quad \widehat{A}^{ridge} := (X^\top X + \lambda_1 I)^{-1} X^\top Y, \tag{8}$$

where $(\cdot)^\dagger$ denotes the Moore-Penrose pseudo inverse of a matrix, making $\widehat{A}^{lsq}$ the minimum-norm solution to the least square problem (3). We run the gradient descent algorithm with these two initial estimates, and choose the estimated $A$ that gives a smaller objective.

## 2.2 Sparse and Lyapunov penalty

Sparse learning for vector auto-regressive models has become a useful tool in many modern time series modeling tasks, where the number $p$ of states in the system is usually larger than the length $T$ of the time series. For example, an important problem in computational biology is to understand the progression of certain biological processes from some measurements, such as temporal gene expression data.

Using an idea similar to (6), we estimate $A$ by

$$\min_A \quad \frac{1}{2} \| Y - XA \|_F^2 + \frac{\lambda_2}{4} \| A^\top \widehat{Q} A + \sigma^2 I - \widehat{Q} \|_F^2, \tag{9}$$
$$\text{s.t.} \quad \| A \|_1 \leq \lambda_1.$$

Instead of adding a sparse penalty on $A$ to the objective function, we impose a constraint on the $\ell_1$ norm of $A$. Both the penalty and the constraint formulations have been considered in the sparse learning literature, and shown to be equivalent in the case of a convex objective. Here we choose the constraint formulation because it can be solved by a simple projected gradient descent method. On the contrary, the penalty formulation leads to a non-smooth and non-convex optimization problem, which is difficult to solve with standard methods for sparse learning. In particular, the soft-thresholding-based coordinate descent method for LASSO does not apply due to the Lyapunov regularization term. Moreover, most of the common methods for non-smooth optimization, such as bundle methods, solve convex problems and need non-trivial modification in order to handle non-convex problems [14].

Let $J(A)$ denote the objective function in (9) and $A^{(k)}$ denote the intermediate solution at the $k$-th iteration. Our projected gradient method updates $A^{(k)}$ to $A^{(k+1)}$ by the following rule:

$$A^{(k+1)} \leftarrow \Pi(A^{(k)} - \eta^{(k)} \nabla J(A^{(k)})), \tag{10}$$

where $\eta^{(k)} > 0$ denotes a proper step size, $\nabla J(A^{(k)})$ denotes the gradient of $J(\cdot)$ at $A^{(k)}$, and $\Pi(\cdot)$ denotes the projection onto the feasible region $\| A \|_1 \leq \lambda_1$. More precisely, for any $p$-by-$p$ real matrix $V$ we define

$$\Pi(V) := \arg \min_{\| A \|_1 \leq \lambda_1} \| A - V \|_F^2. \tag{11}$$

To compute the projection, we use the efficient $\ell_1$ projection technique given in Figure 2 of [5], whose expected running time is linear in the size of $V$.

For choosing a proper step size $\eta^{(k)}$, we consider the simple and effective *Armijo rule along the projection arc* described in [2]. This procedure is given in Algorithm 1, and the main idea is to ensure a sufficient decrease in the objective value per iteration (13). [2] proved that there always exists $\eta^{(k)} = \beta^{r_k} > 0$ satisfying (13), and every limit point of $\{A^{(k)}\}_{k=0}^\infty$ is a stationary point of (9). In our experiments we set $c = 0.01$ and $\beta = 0.1$, both of which are typical values used in gradient descent. As in the previous section, we need good initializations for the projected gradient descent method. Here we use these two initial estimates:

$$\widehat{A}^{lsq'} := \arg \min_{\| A \|_1 \leq \lambda_1} \| A - \widehat{A}^{lsq} \|_F^2 \quad \text{and} \quad \widehat{A}^{sp} := \arg \min_{\| A \|_1 \leq \lambda_1} \frac{1}{2} \| Y - XA \|_F^2, \tag{12}$$

where $\widehat{A}^{lsq}$ is defined in (8), and then choose the one that leads to a smaller objective value.

---

**Algorithm 1:** Armijo's rule along the projection arc

---

**Input** : $A^{(k)}, \nabla J(A^{(k)}), 0 < \beta < 1, 0 < c < 1.$
**Output**: $A^{(k+1)}$

**1** Find $\eta^{(k)} = \max\{\beta^{r_k} | r_k \in \{0, 1, \ldots\}\}$ such that $A^{(k+1)} := \Pi(A^{(k)} - \eta^{(k)} \nabla J(A^{(k)}))$ satisfies

$$J(A^{(k+1)}) - J(A^{(k)}) \le c \, \text{trace}\left(\nabla J(A^{(k)})^\top (A^{(k+1)} - A^{(k)})\right) \tag{13}$$

---

## 2.3 Robust estimation of covariance matrices

To obtain a good estimator for $A$ using the proposed methods, we need a good estimator for the covariance of the stationary distribution of (1). Given an independent sample $\{\mathbf{z}_i\}_{i=1}^n$ drawn from the stationary distribution, the sample covariance is defined as

$$S := \frac{1}{n-1} \sum_{i=1}^n (\mathbf{z}_i - \bar{\mathbf{z}})^\top (\mathbf{z}_i - \bar{\mathbf{z}}), \quad \text{where } \bar{\mathbf{z}} := \frac{\sum_{i=1}^n \mathbf{z}_i}{n}. \tag{14}$$

Although unbiased, the sample covariance is known to be vulnerable to outliers, and ill-conditioned when the number of sample points $n$ is smaller than the dimension $p$. Both issues arise in many real world problems, and the latter is particularly common in gene expression analysis. Therefore, researchers in many fields, such as statistics [17, 20, 11], finance [10], signal processing [3, 4], and recently computational biology [15], have investigated robust estimators of covariances. Most of these results originate from the idea of *shrinkage estimators*, which shrink the covariance matrix towards some target covariance with a simple structure, such as a diagonal matrix. It has been shown in, e.g., [17, 10] that shrinking the sample covariance can achieve a smaller mean-squared error (MSE). More specifically, [10] considers the following linear shrinkage:

$$\widehat{Q} = (1 - \alpha)S + \alpha F \tag{15}$$

for $0 < \alpha < 1$ and some target covariance $F$, and derive a formula for the optimal $\alpha$ that minimizes the mean-squared error:

$$\alpha^* := \arg \min_{0 \le \alpha \le 1} \mathbb{E}(\|\widehat{Q} - Q\|_F^2), \tag{16}$$

which involves unknown quantities such as true covariances of $S$. [15] proposed to estimate $\alpha^*$ by replacing all the population quantities appearing in $\alpha^*$ by their unbiased empirical estimates, and derived the resulting estimator $\widehat{\alpha}^*$ for several types of target $F$. For the experiments in this paper we use the estimator proposed in [15] with the following $F$:

$$F_{ij} = \begin{cases} S_{ij}, & \text{if } i = j, \\ 0 & \text{otherwise}, \end{cases} \quad 1 \le i, j \le p. \tag{17}$$

Denoting the sample correlation matrix as $R$, we give the final estimator $\widehat{Q}$ (Table 1 in [15]) below:

$$\widehat{Q}_{ij} := \begin{cases} S_{ij}, & \text{if } i = j, \\ \widehat{R}_{ij}\sqrt{S_{ii}S_{jj}} & \text{otherwise}, \end{cases} \quad \widehat{R}_{ij} := \begin{cases} 1, & \text{if } i = j, \\ R_{ij}\min(1, \max(0, 1 - \widehat{\alpha}^*)) & \text{otherwise}, \end{cases} \tag{18}$$

$$\widehat{\alpha}^* := \frac{\sum_{i \ne j} \widehat{\text{Var}}(R_{ij})}{\sum_{i \ne j} R_{ij}^2} = \frac{\sum_{i \ne j} \frac{n}{(n-1)^3} \sum_{k=1}^n (w_{kij} - \bar{w}_{ij})^2}{\sum_{i \ne j} R_{ij}^2}, \tag{19}$$

where

$$w_{kij} := (\tilde{\mathbf{z}}_k)_i (\tilde{\mathbf{z}}_k)_j, \qquad \bar{w}_{ij} := \frac{\sum_{k=1}^n w_{kij}}{n}, \tag{20}$$

and $\{\tilde{\mathbf{z}}_i\}_{i=1}^n$ are *standardized* non-sequence samples.

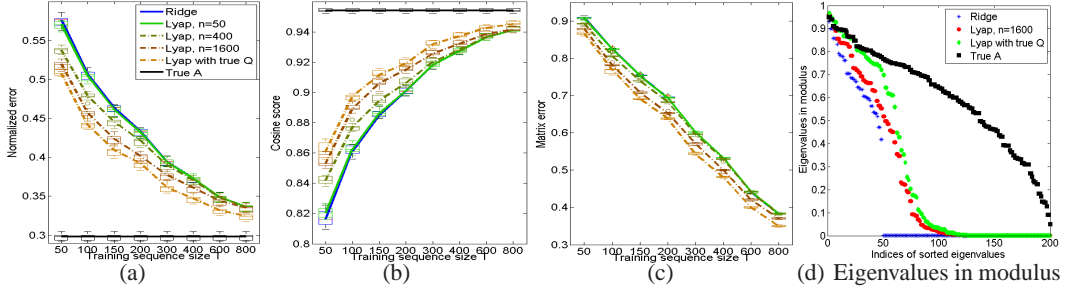

(a)              (b)              (c)          (d) Eigenvalues in modulus

Figure 2: Testing performances and eigenvalues in modulus for the dense model

## 3 Experiments

To evaluate the proposed methods, we conduct experiments on synthetic and video data. In both sets of experiments we use the following two performance measures for a learnt model $\widehat{A}$:

$$\text{Normalized error:} \quad \frac{1}{T-1} \sum_{t=1}^{T-1} \frac{\|\mathbf{x}^{t+1} - \mathbf{x}^t \widehat{A}\|^2}{\|\mathbf{x}^{t+1} - \mathbf{x}^t\|^2}.$$

$$\text{Cosine score:} \quad \frac{1}{T-1} \left| \sum_{t=1}^{T-1} \frac{(\mathbf{x}^{t+1} - \mathbf{x}^t)^\top (\mathbf{x}^t \widehat{A} - \mathbf{x}^t)}{\|\mathbf{x}^{t+1} - \mathbf{x}^t\| \|\mathbf{x}^t \widehat{A} - \mathbf{x}^t\|} \right|.$$

To give an idea of how a good estimate $\widehat{A}$ would perform under these two measures, we point out that a constant prediction $\hat{\mathbf{x}}^{t+1} = \mathbf{x}^t$ leads to a normalized error of 1, and a random-walk prediction $\hat{\mathbf{x}}^{t+1} = \mathbf{x}^t + \boldsymbol{\epsilon}^{t+1}$, $\boldsymbol{\epsilon}^{t+1}$ being a white-noise process, results in a nearly-zero cosine score. Thus, when the true model is more than a simple random walk, a good estimate $\widehat{A}$ should achieve a normalized error much smaller than 1 and a cosine score way above 0. We also note that the cosine score is upper-bounded by 1. In experiments on synthetic data we have the true transition matrix $A$, so we consider a third criterion, the matrix error: $\|\widehat{A} - A\|_F / \|A\|_F$.

In all our experiments, we have a training sequence, a testing sequence, and a non-sequence sample. To choose the hyper-parameters $\lambda_1$, $\lambda_2$ and $\sigma^2$, we split the training sequence into two halves and use the second half as the validation sequence. Once we find the best hyper-parameters according to the validation performance, we train a model on the full training sequence and predict on the testing sequence. For $\lambda_1$ and $\lambda_2$, we adopt the usual grid-search scheme with a suitable range of values. For $\sigma^2$, we observe that (2) implies $\widehat{Q} - \sigma^2 I$ should be positive semidefinite, and thus search the set $\{0.9^j \min_i \lambda_i(\widehat{Q}) \mid 1 \le j \le 3\}$. In most of our experiments, we find that the proposed methods are much less sensitive to $\sigma^2$ than to $\lambda_1$ and $\lambda_2$.

### 3.1 Synthetic Data

We consider the following two VAR models with a Gaussian white noise process $\boldsymbol{\epsilon}^t \sim \mathcal{N}(\mathbf{0}, I)$.

$$\text{Dense Model:} \quad A = \frac{0.95M}{\max(|\lambda_i(M)|)}, M_{ij} \sim \mathcal{N}(0, 1), 1 \le i, j \le 200.$$

$$\text{Sparse Model:} \quad A = \frac{0.95(M \odot B)}{\max(|\lambda_i(M \odot B)|)}, M_{ij} \sim \mathcal{N}(0, 1), B_{ij} \sim \text{Bern}(1/8), 1 \le i, j \le 200,$$

where $\text{Bern}(h)$ is the Bernoulli distribution with success probability $h$, and $\odot$ denotes the entrywise product of two matrices. By setting $h = 1/8$, we make the sparse transition matrix $A$ have roughly $40000/8 = 5000$ non-zero entries. Both models are stable, and the stationary distribution for each model is a zero-mean Gaussian. We obtain the covariance $Q$ of each stationary distribution by solving the Lyapunov equation (2). For a single experiment, we generate a training sequence and a testing sequence, both initialized from the stationary distribution, and draw a non-sequence sample independently from the stationary distribution. We set the length of the testing sequence to be

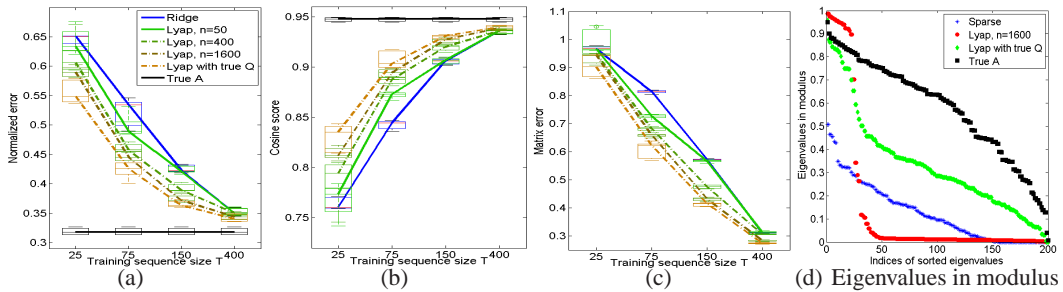

Figure 3: Testing performances and eigenvalues in modulus for the sparse model

800, and vary the training sequence length $T$ and the non-sequence sample size $n$: for the dense model, $T \in \{50, 100, 150, 200, 300, 400, 600, 800\}$ and $n \in \{50, 400, 1600\}$; for the sparse model, $T \in \{25, 75, 150, 400\}$ and $n \in \{50, 400, 1600\}$. Under each combination of $T$ and $n$, we compare the proposed Lyapunov penalization method with the baseline approach of penalized least square, which uses only the sequence data. To investigate the limit of the proposed methods, we also use the true $Q$ for the Lyapunov penalization. We run 10 such experiments for the dense model and 5 for the sparse model, and report the overall performances of both the proposed and the baseline methods.

### 3.1.1 Experimental results for the dense model

We give boxplots of the three performance measures in the 10 experiments in Figures 2(a) to 2(c). The ridge regression approach and the proposed Lyapunov penalization method (6) are abbreviated as Ridge and Lyap, respectively. For normalized error and cosine score, we also report the performance of the true $A$ on testing sequences.

We observe that Lyap improves over Ridge more significantly when the training sequence length $T$ is small ($\leq 200$) and the non-sequence sample size $n$ is large ($\geq 400$). When $T$ is large, Ridge already performs quite well and Lyap does not improve the performance much. But with the true stationary covariance $Q$, Lyap outperforms Ridge significantly for all $T$. When $n$ is small, the covariance estimate $\widehat{Q}$ is far from the true $Q$ and the Lyapunov penalty does not provide useful information about $A$. In this case, the value of $\lambda_2$ determined by the validation performance is usually quite small (0.5 or 1) compared to $\lambda_1$ (256), so the two methods perform similarly on testing sequences. We note that if instead of the robust covariance estimate in (18) and (19) we use the sample covariance, the performance of Lyap can be marginally worse than Ridge when $n$ is small. A precise statement on how the estimation error in $Q$ affects $\widehat{A}$ is worth studying in the future. As a qualitative assessment of the estimated transition matrices, in Figure 2(d) we plot the eigenvalues in modulus of the true $A$ and the $\widehat{A}$'s obtained by different methods when $T = 50$ and $n = 1600$. The eigenvalues are sorted according to their modulus. Both Ridge and Lyap severely under-estimate the eigenvalues in modulus, but Lyap preserves the spectrum much better than Ridge.

### 3.1.2 Experimental results for the sparse model

We give boxplots of the performance measures in the 5 experiments in Figures 3(a) to 3(c), and the eigenvalues in modulus of the true $A$ and some $\widehat{A}$'s in Figure 3(d). The sparse least-square method and the proposed method (9) are abbreviated as Sparse and Lyap, respectively.

We observe the same type of improvement as in the dense model: Lyap improves over Sparse more significantly when $T$ is small and $n$ is large. But the largest improvement occurs when $T = 75$, not the shortest training sequence length $T = 25$. A major difference lies in the impact of the Lyapunov penalization on the spectrum of $\widehat{A}$, as revealed in Figure 3(d). When $T$ is as small as 25, the sparse least-square method shrinks all the eigenvalues but still keep most of them non-zero, while Lyap with a non-sequence sample of size 1600 over-estimates the first few largest eigenvalues in modulus but shrink the rest to have very small modulus. In contrast, Lyap with the true $Q$ preserves the spectrum much better. We may thus need an even better covariance estimate for the sparse model.

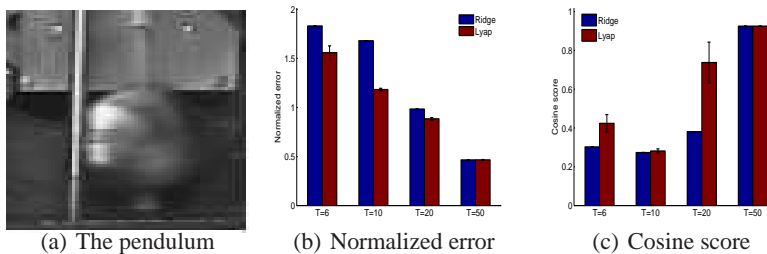

| (a) The pendulum | (b) Normalized error | (c) Cosine score |

Figure 4: Results on the pendulum video data

## 3.2  Video Data

We test our methods using a video sequence of a periodically swinging pendulum[3], which consists of 500 frames of 75-by-80 grayscale images. One such frame is given in Figure 4(a) The period is about 23 frames. To further reduce the dimension we take the second-level Gaussian pyramids, resulting in images of size 9-by-11. We then treat each reduced image as a 99-dimensional vector, and normalize each dimension to be zero-mean and standard deviation 1. We analyze this sequence with a 99-dimensional first-order VAR model. To check whether a VAR model is a suitable choice, we estimate a transition matrix from the first 400 frames by ridge regression while choosing the penalization parameter on the next 50 frames, and predict on the last 50 frames. The best penalization parameter is 0.0156, and the testing normalized error and cosine score are 0.33 and 0.97, respectively, suggesting that the dynamics of the video sequence is well-captured by a VAR model.

We compare the proposed method (6) with the ridge regression for two lengths of the training sequence: $T \in \{6, 10, 20, 50\}$, and treat the last 50 frames as the testing sequence. For both methods, we split the training sequence into two halves and use the second half as a validation sequence. For the proposed method, we simulate a non-sequence sample by randomly choosing 300 frames from between the $(T+1)$-st frame and the 450-th frame without replacement. We repeat this 10 times.

The testing normalized errors and cosine scores of both methods are given in Figures 4(b) and 4(c). For the proposed method, we report the mean performance measures over the 10 simulated non-sequence samples with standard deviation. When $T \leq 20$, which is close to the period, the proposed method outperforms ridge regression very significantly except when $T = 10$ the cosine score of Lyap is barely better than Ridge. However, when we increase $T$ to 50, the difference between the two methods vanishes, even though there is still much room for improvement as indicated by the result of our model sanity check before. This may be due to our use of dependent data as the non-sequence sample, or simply insufficient non-sequence data. As for $\lambda_1$ and $\lambda_2$, their values decrease respectively from 512 and 2,048 to less than 32 as $T$ increases, but since we fix the amount of non-sequence data, the interaction between their value changes is less clear than on the synthetic data.

## 4  Conclusion

We propose to improve penalized least-square estimation of VAR models by incorporating non-sequence data, which are assumed to be samples drawn from the stationary distribution of the underlying VAR model. We construct a novel penalization term based on the discrete-time Lyapunov equation concerning the covariance (estimate) of the stationary distribution. Preliminary experimental results demonstrate that our methods can improve significantly over standard penalized least-square methods when there are only few sequence data but abundant non-sequence data and when the model assumption is valid. In the future, we would like to investigate the impact of $\widehat{Q}$ on $\widehat{A}$ in a precise manner. Also, we may consider noise processes $\epsilon^t$ with more general covariances, and incorporate the noise covariance estimation into the proposed Lyapunov penalization scheme. Finally and the most importantly, we aim to apply the proposed methods to real scientific time series data and provide a more effective tool for those modelling tasks.

## Footnotes

[1] In several disciplines, such as social and medical sciences, the former is usually referred to as a *longitudinal study*, while the latter is similar to what is called a *cross-sectional study*.

[2]If we further require $A$ to be symmetric, (2) would be a simplified *Continuous-time Algebraic Riccati Equation*, which has a unique solution under some conditions (c.f. [1]).

[3]A similar video sequence has been used in [16].

# References

[1] P. Antsaklis and A. Michel. *Linear systems*. Birkhauser, 2005. 2

[2] D. P. Bertsekas. *Nonlinear Programming*. Athena Scientific, Belmont, MA 02178-9998, second edition, 1999. 4

[3] Y. Chen, A. Wiesel, Y. C. Eldar, and A. O. Hero. Shrinkage algorithms for mmse covariance estimation. *IEEE Transactions on Signal Processing*, 58:5016–5029, 2010. 5

[4] Y. Chen, A. Wiesel, and A. O. Hero. Robust shrinkage estimation of high-dimensional covariance matrices. Technical report, arXiv:1009.5331v1 [stat.ME], September 2010. 5

[5] J. Duchi, S. Shalev-Shwartz, Y. Singer, and T. Chandra. Efficient projections onto the $\ell_1$-ball for learning in high dimensions. In *Proceedings of the 25th International Conference on Machine Learning*, pages 272–279, 2008. 4

[6] A. Gupta and Z. Bar-Joseph. Extracting dynamics from static cancer expression data. *IEEE/ACM Transactions on Computational Biology and Bioinformatics*, 5:172–182, 2008. 2

[7] J. Hamilton. *Time series analysis*. Princeton Univ Pr, 1994. 2

[8] T.-K. Huang and J. Schneider. Learning linear dynamical systems without sequence information. In *Proceedings of the 26th International Conference on Machine Learning*, pages 425–432, 2009. 2

[9] T.-K. Huang, L. Song, and J. Schneider. Learning nonlinear dynamic models from non-sequenced data. In *Proceedings of the 13th International Conference on Artificial Intelligence and Statistics*, 2010. 2

[10] O. Ledoit and M. Wolf. Improved estimation of the covariance matrix of stock returns with an application to portfolio selection. *Journal of Empirical Finance*, 10:603–621, 2003. 5

[11] O. Ledoit and M. Wolf. A well-conditioned estimator for large-dimensional covariance matrices. *Journal of Multivariate Analysis*, 88:365–411, 2004. 5

[12] A. Lozano, N. Abe, Y. Liu, and S. Rosset. Grouped graphical granger modeling for gene expression regulatory networks discovery. *Bioinformatics*, 25(12):i110, 2009. 1

[13] T. C. Mills. *The Econometric Modelling of Financial Time Series*. Cambridge University Press, second edition, 1999. 1

[14] D. Noll, O. Prot, and A. Rondepierre. A proximity control algorithm to minimize nonsmooth and nonconvex functions. *Pacific Journal of Optimization*, 4(3):569–602, 2008. 4

[15] J. Schäfer and K. Strimmer. A shrinkage approach to large-scale covariance matrix estimation and implications for functional genomics. *Statistical Applications in Genetics and Molecular Biology*, 4, 2005. 5

[16] S. M. Siddiqi, B. Boots, and G. J. Gordon. Reduced-rank hidden Markov models. In *Proceedings of the 13th International Conference on Artificial Intelligence and Statistics*, 2010. 8

[17] C. Stein. Estimation of a covariance matrix. In *Rietz Lecture, 39th Annual Meeting, Atlanta, GA*, 1975. 5

[18] R. S. Tsay. *Analysis of financial time series*. Wiley-Interscience, 2005. 1

[19] B. P. Tu, A. Kudlicki, M. Rowicka, and S. L. McKnight. Logic of the yeast metabolic cycle: Temporal compartmentalization of cellular processes. *Science*, 310(5751):1152–1158, 2005. 1

[20] R. Yang and J. O. Berger. Estimation of a covariance matrix using the reference prior. *Annals of Statistics*, 22:1195–1211, 1994. 5

